# Differential Use of Implicit Negative Evidence in Generative and Discriminative Language Learning

**Anne S. Hsu**   **Thomas L. Griffiths**
Department of Psychology
University of California, Berkeley
Berkeley, CA 94720
{showen,tom_griffiths}@berkeley.edu

## Abstract

A classic debate in cognitive science revolves around understanding how children learn complex linguistic rules, such as those governing restrictions on verb alternations, without negative evidence. Traditionally, formal learnability arguments have been used to claim that such learning is impossible without the aid of innate language-specific knowledge. However, recently, researchers have shown that statistical models are capable of learning complex rules from only positive evidence. These two kinds of learnability analyses differ in their assumptions about the distribution from which linguistic input is generated. The former analyses assume that learners seek to identify grammatical sentences in a way that is robust to the distribution from which the sentences are generated, analogous to discriminative approaches in machine learning. The latter assume that learners are trying to estimate a generative model, with sentences being sampled from that model. We show that these two learning approaches differ in their use of implicit negative evidence – the absence of a sentence – when learning verb alternations, and demonstrate that human learners can produce results consistent with the predictions of both approaches, depending on how the learning problem is presented.

## 1 Introduction

Languages have a complex structure, full of general rules with idiosyncratic exceptions. For example, the causative alternation in English allows a class of verbs to take both the transitive form, "I opened the door", and the intransitive form, "The door opened". With other verbs, alternations are restricted, and they are grammatical in only one form. For example, "The rabbit disappeared" is grammatical whereas "I disappeared the rabbit" is ungrammatical. There is a great debate over how children learn language, related to the infamous "poverty of the stimulus" argument [1, 2, 3, 4]. A central part of the debate arises from the fact that a child mostly learns language only by hearing adults speak grammatical sentences, known as *positive evidence*. Children are believed to learn language mostly from positive evidence because research has found that children rarely receive indications from parents that a sentence is not grammatical, and they ignore these indications when they do recieve them. An explicit indication that a sentence is not grammatical is known as *negative evidence* [5, 6, 7]. Yet, speaking a language speaking involves the generalization of linguistic patterns into novel combinations of phrases that have never been heard before. This presents the following puzzle: How do children eventually learn that certain novel linguistic generalizations are not allowed if they are not explicitly told? There have been two main lines of analyses addressing this question. These analyses have taken two different perspectives on the basic task involved in language learning, and have yielded quite different results.

One perspective is that language is acquired by learning rules for identifying grammatically acceptable and unacceptable sentences in a way that is robust to the actual distribution of observed

sentences. From this perspective, Gold's theorem [8] asserts that languages with infinite recursion, such as most human languages, are impossible to learn from positive evidence alone. In particular, linguistic exceptions, such as the restrictions on verb alternations mentioned above, are cited as being impossible to learn empirically. More recent analyses yield similar results, while making weaker assumptions about the desired outcome of learning (for a review, see [9]). In light of this, it has been argued that child language learning abilities can only be explained by the presence of innate knowledge specific to language [3, 4, 10].

On the other side of the debate, results indicating that relatively sophisticated linguistic representations such as probabilistic context-free grammars can be learned from positive evidence have been obtained by viewing language acquisition as a process of forming a probabilistic model of the linguistic input, under the assumption that the observed data are sampled from this model [11, 12, 13]. In addition to these general theoretical results, statistical learning models have been shown to be capable of learning exceptions in language from positive examples only in a variety of domains, including verb alternations [14, 15, 16, 17, 18, 19]. Furthermore, previous experimental work has shown that humans are capable of learning linguistic exceptions in an artificial language without negative evidence [20], bearing out the predictions of some of these models.

One key difference between these two perspectives on learning is in the assumptions that they make about how observed sentences are generated. In the former approach, the goal is to learn to identify grammatical sentences without making assumptions about the distribution from which they are drawn. In the latter approach, the goal is to learn a probability distribution over sentences, and the observed sentences are assumed to be drawn from that distribution. This difference is analogous to the distinction between discriminative and generative models in machine learning (e.g., [21]). The stronger distributional assumptions made in the generative approach result in a less robust learner, but make it possible to learn linguistic exceptions without negative evidence. In particular, generative models can exploit the "implicit negative evidence" provided by the absence of a sentence: the assumption that sentences are generated from the target probability distribution means that not observing a sentence provides weak evidence that it does not belong to the language. In contrast, discriminative models that seek to learn a function for labelling sentences as grammatical or ungrammatical are more robust to the distribution from which the sentences are drawn, but their weaker assumptions about this distribution mean that they are unable to exploit implicit negative evidence.

In this paper, we explore how these two different views of learning are related to human language acquisition. Here we focus on the task of learning an artifical language containing both alternating and non-alternating verbs. Our goal is to use modeling and human experiments to demonstrate that the opposing conclusions from the two sides of the language acquisition debate can be explained by a difference in learning approach. We compare the learning performance of a hierarchical Bayesian model [15], which takes a generative approach, with a logistic regression model, which takes a discriminative approach. We show that without negative evidence, the generative model will judge a verb structure that is absent in the input to be ungrammatical, while the discriminative model will judge it to be grammatical. We then conduct an experiment designed to encourage human participants to adopt either a generative or discriminative language learning perspective. The experimental results indicate that human learners behave in accordance with model predictions: absent verb structures are rejected as ungrammatical under a generative learning perspective and accepted as grammatical under a discriminative one. Our modeling comparisons and experimental results contribute to the language acquisition debate in the following ways: First, our results lend credence to conclusions from both sides of the debate by showing that linguistic exceptions appear either unlearnable or learnable, depending on the learning perspective. Second, our results indicate that the opposing conclusions about learnability can indeed be attributed to whether one assumes a discriminative or a generative learning perspective. Finally, because our generative learning condition is much more similar to actual child language learning, our results lend weight to the argument that children can learn language empirically from positive input.

## 2  Models of language learning: Generative and discriminative

Generative approaches seek to infer the probability distribution over sentences that characterizes the language, while discriminative models seek to identify a function that indicates whether a sentence is grammatical. General results exist that characterize the learnability of languages from these two

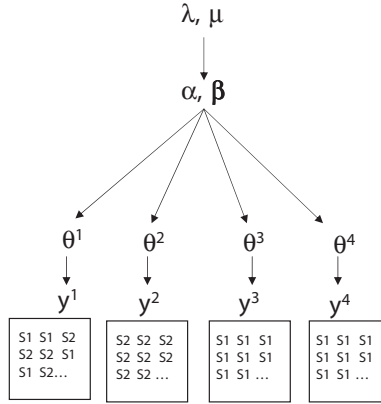

Figure 1: A hierarchical Bayesian model for learning verb alternations. Figure adapted from [15].

perspectives, but there are few direct comparisons of generative and discriminative approaches to the same specific language learning situation. Here, we compare a simple generative and discriminative model's predictions of how implicit negative evidence is used to learn verb alternations.

## 2.1 Generative model: Hierarchical Bayes

In the generative model, the problem of learning verb alternations is formulated as follows. Assume we have a set of $m$ verbs, which can occur in up to $k$ different sentence structures. Restricting ourself to positive examples for the moment, we observe a total of $n$ sentences $x_1, \ldots x_n$. The $n^i$ sentences containing verb $i$ can be summarized in a $k$-dimensional vector $\boldsymbol{y}^i$ containing the verb occurrence frequency in each of the $k$ sentence structures. For example if we had three possible sentence structure types and verb $i$ occurred in the first type two times, the second type four times and the third type zero times, $\boldsymbol{y}^i$ would be $[2, 4, 0]$ and $n^i$ would be 6.

We model these data using a hierarchical Bayesian model (HBM) originally introduced in [15], also known to statisticians as a Dirichlet-Multinomial model [22]. In statistical notation the HBM is

$$
\begin{array}{llllll}
\boldsymbol{\theta}^i & \sim & \text{Dirichlet}(\alpha\boldsymbol{\beta}) & \alpha & \sim & \text{Exponential}(\lambda) \\
\boldsymbol{y}^i | n^i & \sim & \text{Multinomial}(\boldsymbol{\theta}^i) & \boldsymbol{\beta} & \sim & \text{Dirichlet}(\boldsymbol{\mu})
\end{array}
$$

where $\boldsymbol{y}^i$ is the data (i.e. the observed frequency of different grammatical sentence structures for verb $i$) given $n^i$ occurrences of that verb, as summarized above. $\boldsymbol{\theta}^i$ captures the distribution over sentence structures associated with verb $i$, assuming that sentences are generated independently and structure $k$ is generated with probability $\theta_k^i$. The hyperparameters $\alpha$ and $\boldsymbol{\beta}$ represent generalizations about the kinds of sentence structures that typically occur. More precisely, $\boldsymbol{\beta}$ represents the distribution of sentence structures across all verbs, with $\beta_k$ being the mean probability of sentence structure $k$, while $\alpha$ represents the extent to which verbs tends to appear in only one sentence structure type.

In this model, the number of verbs and the number of possible sentence structures are both fixed. The hyperparameters $\alpha$ and $\boldsymbol{\beta}$ are learned, and the prior on these hyperparameters is fixed by setting $\lambda = 1$ and $\boldsymbol{\mu} = 1$ for all $i$. This prior asserts a weak expectation that the range of $\alpha$ and $\boldsymbol{\beta}$ do not contain extreme values. The model is fit to the data by computing the posterior distribution $p(\boldsymbol{\theta^i}|\boldsymbol{y^i}) = \int_{\alpha,\boldsymbol{\beta}} p(\boldsymbol{\theta^i}|\alpha, \boldsymbol{\beta}, y)p(\alpha, \boldsymbol{\beta}|y) \, d\alpha \, d\boldsymbol{\beta}$. The posterior can be estimated using a Markov Chain Monte Carlo (MCMC) algorithm. Following [15], we use Gaussian proposals on $\log(\alpha)$, and draw proposals for $\boldsymbol{\beta}$ from a Dirichlet distribution with the current $\boldsymbol{\beta}$ as its mean.

## 2.2 Discriminative model: Logistic regression

For our discriminative model we use logistic regression. A logistic regression model can be used to learn a function that classifies observations into two classes. In the context of language learning, the observations are sentences and the classification problem is deciding whether each sentence is grammatical. As above, we observe $n$ sentences, $x_1, \ldots x_n$, but now each sentence $x_j$ is associated

with a variable $c_j$ indicating whether the sentence is grammatical ($c_j = +1$) or ungrammatical ($c_j = -1$). Each sentence is associated with a feature vector $\mathbf{f}(x_j)$ that uses dummy variables to encode the verb, the sentence structure, and the interaction of the two (ie. each sentence's particular verb and sentence structure combination). With $m$ verbs and $k$ sentence structures, this results in $m$ verb features, $k$ sentence structure features, and $mk$ interaction features, each of which take the value 1 when they match the sentence and 0 when they do not. For example, a sentence containing the second of four verbs in the first of three sentence structures would be encoded with the binary feature vector 0100100000100000000.

The logistic regression model learns which features of sentences are predictive of grammaticality. This is done by defining the probability of grammaticality to be

$$p(c_j = +1|x_j, \boldsymbol{w}, b) = 1/(1 + \exp\{-\mathbf{w}^T \mathbf{f}(x_j) - b\}) \tag{1}$$

where $\mathbf{w}$ and $b$ are the parameters of the model. $\boldsymbol{w}$ and $b$ are estimated by maximizing the log likelihood $\sum_{j=1}^{n} \log p(c_j|x_j, \mathbf{w}, b)$. Features for which the likelihood is uninformative (e.g. features that are not observed) have weights that are set to zero.

## 3 Testing the models on an artificial language

To examine the predictions that these two models make about the use of implicit negative evidence in learning verb alternations, we applied them to a simple artificial language based on that used in [20]. This language has four transitive verbs and three possible sentence structures. Three of the verbs only appear in one sentence structure (non-alternating), while one verb appears in two possible sentence structures (alternating). The language consisted of three-word sentences, each containing a subject (N1), object (N2) and verb (V), with the order depending on the particular sentence structure.

### 3.1 Vocabulary

The vocabulary was a subset of that used in [20]. There were three two-syllable nouns, each beginning with a different consonant, referring to three cartoon animals: *blergen* (lion), *nagid* (elephant), *tombat* (giraffe). Noun referents are fixed across participants. The four one-syllable verbs were: *gund*, *flern*, *semz*, and *norg*, corresponding to the four transitive actions: *eclipse*, *push-to-side*, *explode* and *jump on*. While the identity of the nouns and verbs is irrelevant to the models, we developed this language with the intent of also examining human learning, as described below. With human learners, the mapping of verbs to actions was randomly selected for each participant.

### 3.2 Syntax and grammar

In our language of three-word sentences, a verb could appear in 3 different positions (as the 1st, 2nd or 3rd word). We constrained the possible sentences such that the subject, N1, always appeared before the object, N2. This leaves us with three possible sentence structures, S1,S2, and S3, each of which corresponded to one of the following word orders: N1-N2-V, N1-V-N2 and V-N1-N2. In our experiment, the mapping from sentence structure to word order was randomized among participants. For example, S1 might correspond to N1-N2-V for one participant or it might correspond to V-N1-N2 for another participant. There was always one sentence structure, which we denote S3, that was never grammatical for any of the verbs. For S1 and S2, grammaticality varied depending on the verb. We designed our language to have 1 alternating verb and 3 non-alternating verbs. One of the three non-alternating verbs was only grammatical in S1. The other two non-alternating verbs were only grammatical in S2. For example, let's consider the situation where S1 is N1-V-N2, S2 is N1-N2-V and S3 is V-N1-N2. If *flern* was an alternating verb, both *nagid flern tombat* and *nagid tombat flern* would be allowed. If *semz* was non-alternating, and only allowed in S2, *nagid tombat semz* would be grammatical and *nagid tombat semz* would be ungrammatical. In this example, *flern nagid tombat* and *semz nagid tombat* are both ungrammatical. The language is summarized in Table 1.

### 3.3 Modeling results

The generative hierarchical Bayesian model and the discriminative logistic regression model outlined in the previous section were applied to a corpus of sentences generated from this language.

|       | Sentence Structure | | |
|-------|------|------|------|
| Verb  | S1   | S2   | S3   |
| V1    | +(9) | +(9) | -(9) |
| V2    | -(3) | +(18)| -(3) |
| V3    | +(18)| -(3) | -(3) |
| V4    | +(18)| ?(0) | -(6) |

Table 1: Grammaticality of verbs. + and - indicate grammatical and ungrammatical respectively, while ? indicates that grammaticality is underdetermined by the data. The number in parentheses is the frequency with which each sentence was presented to model and human learners in our experiment. Verb V4 was never shown in sentence structure S2. Grammaticality predictions for sentences containing this verb were used to explore the interpretation of implicit negative evidence.

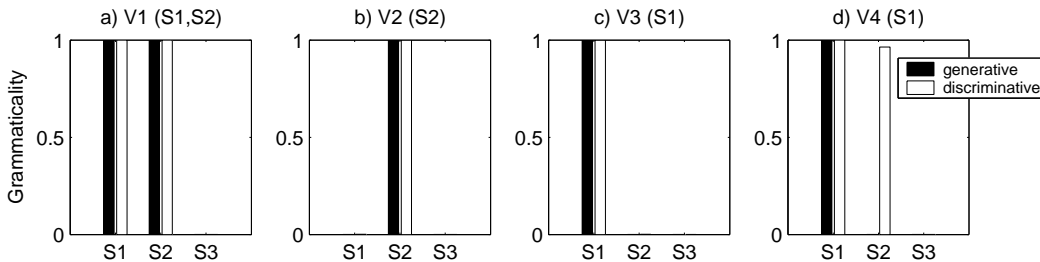

Figure 2: Predicted grammaticality judgments from generative and discriminative models. In parentheses next to the verb index in the title of each plot is the sentence structure(s) that were shown to be grammatical for that verb in the training corpus.

The frequencies of each verb and sentence structure combination are also shown in Table 1. We were particularly interested in the predictions that the two models made about the grammaticality of verb V4 in sentence structure S2, since this combination of verb and sentence structure never occurs in the data. As a consequence, a generative learner receives implicit negative evidence that S2 is not grammatical for V4, while a discriminative learner receives no information.

We trained the HBM on the grammatical instances of the sentences, using 10,000 iterations of MCMC. The results indicate that V1 is expected to occur in both S1 and S2 50% of the time, while all other verbs are expected to occur 100% of the time in the one sentence structure for which they are grammatical, accurately reflecting the distribution in our language input. Predictions for grammaticality are extracted from the HBM model as follows: The $i$th verb is grammatical in sentence structure $k$ if the probability of sentence structure $k$, $\theta_k^i$ is greater than or equal to $\epsilon$ and ungrammatical otherwise, where $\epsilon$ is a small number. Theoretically, $\epsilon$ should be set so that any sentence observed once will be considered grammatical. Here, posterior values of $\theta_k^i$ were highly peaked about 0.5 for V1 in S1 and S2, and either 0 or 1 for other verb and sentence structure combinations, resulting in clear grammaticality predictions. These are shown in Figure 2. Critically, the model predicts that V4 in S2 is not grammatical.

Logistic regression was performed using all sentences in our corpus, both grammatical and ungrammatical. Predictions for grammaticality from the logistic regression model were read out directly from $p(c_j = +1|x_j, \mathbf{w}, b)$. The results are shown in Figure 2. While the model has not seen V4 in S2, and has consequently not estimated a weight for the feature that uniquely identifies this sentence, it has seen 27 grammatical and 3 ungrammatical instances of S2, and 18 grammatical and 6 ungrammatical instances of V4, so it has learned positive weights for both of these features of sentences. As a consequence, it predicts that V4 in S2 is grammatical.

## 4 Generative and discriminative learning in humans

The simulations above illustrate how generative and discriminative approaches to language learning differ in their treatment of implicit negative evidence. This raises the question of whether a similar difference can be produced in human learners by changing the nature of the language learning task. We conducted an experiment to explore whether this is the case.

In our experiment, participants learned the artificial language used to generate the model predictions in the previous section by watching computer animated scenes accompanied by spoken and written sentences describing each scene. Participants were also provided with information about whether the sentence was grammatical or ungrammatical. Participants were assigned to one of two conditions, which prompted either generative or discriminative learning. Participants in both conditions were exposed to exactly the same sentences and grammaticality information. The two conditions differed only in how grammaticality information presented.

## 4.1 Participants

A total of 22 participants were recruited from the community at the University of California, Berkeley.

## 4.2 Stimuli

As summarized in Table 1, participants viewed each of the 4 verbs 24 times, 18 grammatical sentences and 6 ungrammatical sentences. The alternating verb was shown 9 times each in S1 and S2 and 6 times in S3. The non-alternating verbs were shown 18 times each in their respectively grammatical sentence structures and 3 times each in the 2 ungrammatical structures. Presentation of sentences was ordered as follows: Two chains of sentences were constructed, one grammatical and one ungrammatical. The grammatical chain consisted of 72 sentences (18 for each verb) and the ungrammatical chain consisted of 24 sentences (6 for each verb). For each sentence chain, verbs were presented cyclically and randomized within cycles. For the grammatical chain, V1 occurrences of S1 and S2 were cycled through in semi-random order (verbs V2-V4 appeared grammatically in only one sentence construction). Similarly, for the ungrammatical chain, V2 and V3 cycled semi-randomly through occurrences of S1 and S3 and S2 and S3 respectively (verbs V1 and V4 only appeared ungrammatically in S3). While participants were being trained on the language, presentation of one sentence from the ungrammatical chain was randomly interleaved within every three presentations of sentences from the grammatical chain. Subject-object noun pairs were randomized for each verb across presentations. There were a total of 96 training sentences.

## 4.3 Procedure

Participants in both conditions underwent pre-training trials to acquaint them with the vocabulary. During pre-training they heard and saw each word along with pictures of each noun and scenes corresponding to each verb along with spoken audio of each noun/verb. All words were cycled through three times during pre-training. During the main experiment, all participants were told they were to learn an artificial language. They all saw a series of sentences describing animated scenes where a subject noun performed an action on an object noun. All sentences were presented in both spoken and written form.

### 4.3.1 Generative learning condition

In the generative learning condition, participants were told that they would listen to an adult speaker who was always spoke grammatical sentences and a child speaker who always spoke ungrammatically. Cartoon pictures of either the adult or child speaker accompanied each scene. The child speaker's voice was low-pass filtered to create a believably child-like sound. We hypothesized that participants in this condition would behave similarly to a generative model: they would build a probabilistic representation of the language from the grammatical sentences produced by the adult speaker.

### 4.3.2 Discriminative learning condition

In the discriminative learning condition, participants were presented with spoken and written sentences describing each scene and asked to choose whether each of the presented sentences were grammatical or not. They were assured that only relevant words were used and they only had to figure out if the verb occurred in a grammatical location. Participants then received feedback on their choice. For example, if a participant answered that the sentence was grammatical, they would see either "Yes, you were correct. This sentence is grammatical!" or "Sorry, you were incorrect. This

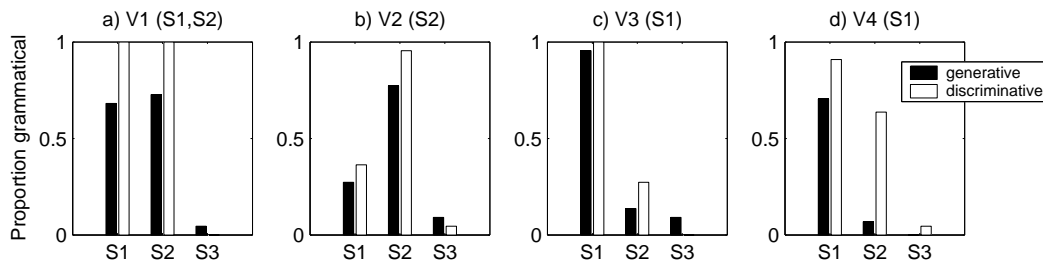

Figure 3: Human grammar judgments, showing proportion grammatical for each sentence structure.

sentence is ungrammatical!" The main difference from the generative condition is that in the discriminative condition, the presented sentences are assumed to be chosen at random, whereas in the generative learning condition, sentences from the adult speaker are assumed to have been sampled from the language distribution. We hypothesized that participants in the discriminative condition would behave similarly to a discriminative model: they would use feedback about both grammatical and ungrammatical sentences to formulate rules about what made sentences grammatical.

### 4.3.3 Testing

After the language learning phase, participants in both conditions were subjected to a grammar test. In this testing phase, participants were shown a series of written sentences and asked to rate the sentence as either grammatical or ungrammatical. Here, all sentences had *blergen* as the subject and *nagid* as the object. All verb-sentence structure combinations were shown twice. Additionally the verb V4 was shown an extra two times in S2 as this was the crucial generalization that we were testing.

Participants also underwent a production test in which they were shown a scene and asked to type in a sentence describing that scene. Because we did not want this to be a memory test, we displayed the relevant verb on the top of the screen. Pictures of all the nouns, with their respective names below, were also available on the bottom of the screen for reference. Four scenes were presented for each verb, using subject-object noun pairs that were cycled through random. Verbs were also cycled through at random.

### 4.4 Results

Our results show that participants in both conditions were largely able to learn much of the grammar structure. Hoewever, there were significant differences between the generative and discriminative conditions (see Figure 3). Most notably, the generative learners overwhelmingly judged verb V4 to be ungrammatical in S2, while the majority of discriminative learners deemed V4 in to be grammatical in S2 (see Figure 3d). This difference between conditions was highly statistically significant by a Pearson's $\chi^2$ test ($\chi^2(1) = 7.28, p = 0.007$). This difference aligned with the difference in the predictions of the HBM (generative) model and the logistic regression (discriminative) model discussed earlier. Our results strongly suggest participants in the generative condition were learning language with a probabilistic perspective that allowed them to learn restrictions on verb alternations by using implicit negative evidence whereas participants in the discriminative condition made sampling assumptions that did not allow them to learn the alternation restriction.

Another difference we found between the two conditions was that discriminative learners were more willing to consider verbs to be alternating (i.e. allow those verbs to be grammatical in two sentence structures.) This is evidenced by the fact that participants in the generative condition rated occurrences of V1 (the alternating verb) in S1 and S2 as grammatical only 68% and 72% of the time. This is because many participants judged V1 to be grammatical in either S1 or S2 and not both. On the other hand, participants in the discriminative condition rated occurrences of V1 in S1 and S2 grammatical 100% of the time (see Figure 3a). Pearson's $\chi^2$ tests for the difference between conditions for grammaticality of V1 in S1 and S2 were marginally significant, with $\chi^2(1) = 4.16, p = .04$ and $\chi^2(1) = 3.47, p = 0.06$ respectively. From post-experiment questioning, we learned that many participants in the generative condition did not think verbs would occur in two possible sentence

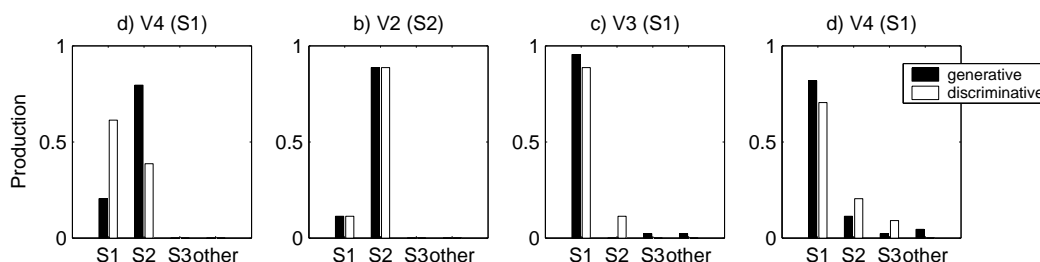

Figure 4: Human production data, showing proportion of productions in each sentence structure.

structures. None of the participants in the discriminative condition were constrained by this assumption. Why the two conditions prompted significantly different prior assumptions about the prevalence of verb alternations will be a question for future research, but is particularly interesting in the context of the HBM, which can learn a prior expressing similar constraints.

Production test results showed that participants tended to use verbs in the sentences structure that they heard them in (see Figure 4). Notably, even though the majority of the learners in the discriminative condition rated verb V4 in S2 as grammatical, only 20% of the productions of V4 were in S2. This is in line with previous results that show that how often a sentence structure is produced is proportional to how often that structure is heard, and rarely heard structures are rarely produced, even if they are believed to be grammatical [20].

## 5 Discussion

We have shown that artificial language learners may or may not learn restrictions on verb alternations, depending on the learning context. Our simulations of generative and discriminative learners made predictions about how these approaches deal with implicit negative evidence, and these predictions were borne out in an experiment with human learners. Participants in both experimental conditions viewed exactly the same sentences and were told whether each sentence was grammatical or ungrammatical. What varied between conditions was the way the the grammaticality information was presented. In the discriminative condition, participants were given yes/no grammaticality feedback on sentences presumed to be sampled at random. Because of the random sampling assumption, the absence of a verb in a given sentence structure did not provide implicit negative evidence against the grammaticality of that construction. In contrast, participants in the generative condition judged the unseen verb-sentence structure to be ungrammatical. This is in line with the idea that they had sought to estimate a probability distribution over sentences, under the assumption that the sentences they observed were drawn from that distribution.

Our simulations and behavioral results begin to clarify the connection between theoretical analyses of language learnability and human behavior. In showing that people learn differently under different construals of the learning problem, we are able to examine how well normal language learning corresponds to the learning behavior we see in these two cases. Participants in our generative condition heard sentences spoken by a grammatical speaker, similar to the way children learn by listening to adult speech. In post-experiment questioning, generative learners also stated that they ignored all negative evidence from the ungrmamatical child speaker, similar to the way children ignore negative evidence in real language acquisition. These observations support the idea that human language learning is better characterized by the generative approach. Establishing this connection to the generative approach helps to identify the strengths and limitations of human language learning, leading to the expectation that human learners can use implicit negative evidence to identify their language, but will not be as robust to variation in the distribution of observed sentences as a discriminative learner might be.

**Acknowledgments.** This work was supported by grant SES-0631518 from the National Science Foundation.

# References

[1] C. L. Baker. Syntactic theory and the projection problem. *Linguistic Inquiry*, 10:533–538, 1979.

[2] C. L. Baker and J. J. McCarthy. *The logical problem of language acquisition*. MIT Press, 1981.

[3] N. Chomsky. *Aspects if the theories of syntax*. MIT Press, 1965.

[4] S. Pinker. *Learnability and Cognition: The acquisition of argument structure*. MIT Press, 1989.

[5] M. Bowerman. The 'No Negative Evidence' Problem: How do children avoid constructing an overly general grammar? In J. Hawkins, editor, *Explaining Language Universals*, pages 73–101. Blackwell, New York, 1988.

[6] R. Brown and C. Hanlon. *Derivational complexity and order of acquisition in child speech*. Wiley, 1970.

[7] G. F. Marcus. Negative evidence in language acquisition. *Cognition*, 46:53–85, 1993.

[8] E. M. Gold. Language identification in the limit. *Information and Control*, 16:447–474, 1967.

[9] M. A. Nowak, N. L. Komarova, and P. Niyogi. Computational and evolutionary aspects of language. *Nature*, 417:611–617, 2002.

[10] S. Crain and L. D. Martin. *An introduction to linguistic theory and language acquisition*. Blackwell, 1999.

[11] D. Angluin. Identifying languages from stochastic examples. Technical Report YALEU/DCS/RR-614, Yale University, Department of Computer Science, 1988.

[12] J. J. Horning. *A study of grammatical inference*. PhD thesis, Stanford University, 1969.

[13] N. Chater and P. Vitanyi. "Ideal learning" of natural language: Positive results about learning from positive evidence. *Journal of Mathematical Psychology*, 51:135–163, 2007.

[14] M. Dowman. Addressing the learnability of verb subcategorizations with Bayesian inference. In *Proceedings of the 22nd Annual Conference of the Cognitive Science Society*, 2005.

[15] D. Kemp, A. Perfors, and J. Tenenbaum. Learning overhypothesis with hierarchical Bayesian models. *Developmental Science*, 10:307–321, 2007.

[16] P. Langley and S. Stromsten. Learning context-free grammars with a simplicity bias. In *Proceedings of the 11th European Conference on Machine Learning*, 2000.

[17] L. Onnis, M. Roberts, and N. Chater. Simplicity: A cure for overgeneralizations in language acquisition? In *Proceedings of the 24th Annual Conference of the Cognitive Science Society*, pages 720–725, 2002.

[18] A. Perfors, J. Tenenbaum, and T. Regier. Poverty of the stimulus: A rational approach? In *Proceedings of the 28th Annual Conference of the Cognitive Science Society*, pages 664–668, 2006.

[19] A. Stolcke. *Bayesian learning of probabilistic language models*. PhD thesis, UC Berkeley, 1994.

[20] E. Wonnacott, E. Newport, and M. Tanenhaus. Acquiring and processing verb argument structure: Distributional learning in a miniature language. *Cognitive Psychology*, 56:165–209, 2008.

[21] A. Y. Ng and M. Jordan. On discriminative vs. generative classifiers: A comparison of logistic regression and naive Bayes. In *Advances in Neural Information Processing Systems 17*, 2001.

[22] A. Gelman, J. B. Carlin, H. S. Stern, and D. B. Rubin. *Bayesian data analysis*. Chapman Hall, 2003.
